# Logistic Normal Priors for Unsupervised Probabilistic Grammar Induction

**Shay B. Cohen    Kevin Gimpel    Noah A. Smith**
Language Technologies Institute
School of Computer Science
Carnegie Mellon University
{scohen,kgimpel,nasmith}@cs.cmu.edu

## Abstract

We explore a new Bayesian model for probabilistic grammars, a family of distributions over discrete structures that includes hidden Markov models and probabilistic context-free grammars. Our model extends the correlated topic model framework to probabilistic grammars, exploiting the logistic normal distribution as a prior over the grammar parameters. We derive a variational EM algorithm for that model, and then experiment with the task of unsupervised grammar induction for natural language dependency parsing. We show that our model achieves superior results over previous models that use different priors.

## 1   Introduction

Unsupervised learning of structured variables in data is a difficult problem that has received considerable recent attention. In this paper, we consider learning *probabilistic grammars*, a class of structure models that includes Markov models, hidden Markov models (HMMs) and probabilistic context-free grammars (PCFGs). Central to natural language processing (NLP), probabilistic grammars are recursive generative models over discrete graphical structures, built out of conditional multinomial distributions, that make independence assumptions to permit efficient exact probabilistic inference.

There has been an increased interest in the use of Bayesian methods as applied to probabilistic grammars for NLP, including part-of-speech tagging [10, 20], phrase-structure parsing [7, 11, 16], and combinations of models [8]. In Bayesian-minded work with probabilistic grammars, a common thread is the use of a Dirichlet prior for the underlying multinomials, because as the conjugate prior for the multinomial, it bestows computational feasibility. The Dirichlet prior can also be used to encourage the desired property of sparsity in the learned grammar [11].

A related widely known example is the *latent Dirichlet allocation* (LDA) model for topic modeling in document collections [5], in which each document's topic distribution is treated as a hidden variable, as is the topic distribution from which each word is drawn.[1] Blei and Lafferty [4] showed empirical improvements over LDA using a logistic normal distribution that permits different topics to correlate with each other, resulting in a *correlated topic model* (CTM). Here we aim to learn analogous correlations such as: a word that is likely to take one kind of argument (e.g., singular nouns) may be likely to take others as well (e.g., plural or proper nouns). By permitting such correlations via the distribution over the

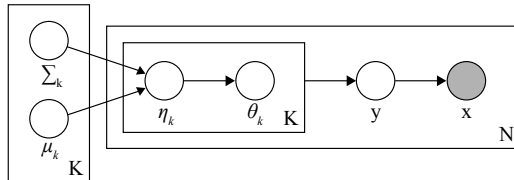

Figure 1: A graphical model for the logistic normal probabilistic grammar. $\mathbf{y}$ is the derivation, $\mathbf{x}$ is the observed string.

parameters, we hope to break independence assumptions typically made about the behavior of different part-of-speech tags.

In this paper, we present a model, in the Bayesian setting, which extends CTM for probabilistic grammars. We also derive an inference algorithm for that model, which is ultimately used to provide a point estimate for the grammar, permitting us to perform fast and exact inference. This is required if the learned grammar is to be used as a component in an application.

The rest of the paper is organized as follows. §2 gives a general form for probabilistic grammars built out of multinomial distributions. §3 describes our model and an efficient variational inference algorithm. §4 presents a probabilistic context-free dependency grammar often used in unsupervised natural language learning. Experimental results showing the competitiveness of our method for estimating that grammar are presented in §5.

## 2   Probabilistic Grammars

A probabilistic grammar defines a probability distribution over a certain kind of structured object (a derivation of the underlying symbolic grammar) explained through step-by-step stochastic process. HMMs, for example, can be understood as a random walk through a probabilistic finite-state network, with an output symbol sampled at each state. PCFGs generate phrase-structure trees by recursively rewriting nonterminal symbols as sequences of "child" symbols (each itself either a nonterminal symbol or a terminal symbol analogous to the emissions of an HMM). Each step or emission of an HMM and each rewriting operation of a PCFG is conditionally independent of the others given a single structural element (one HMM or PCFG state); this Markov property permits efficient inference.

In general, a probabilistic grammar defines the joint probability of a string $\mathbf{x}$ and a grammatical derivation $\mathbf{y}$:

$$p(\mathbf{x}, \mathbf{y} \mid \boldsymbol{\theta}) = \prod_{k=1}^{K} \prod_{i=1}^{N_k} \theta_{k,i}^{f_{k,i}(\mathbf{x},\mathbf{y})} = \exp \sum_{k=1}^{K} \sum_{i=1}^{N_k} f_{k,i}(\mathbf{x}, \mathbf{y}) \log \theta_{k,i} \tag{1}$$

where $f_{k,i}$ is a function that "counts" the number of times the $k$th distribution's $i$th event occurs in the derivation. The parameters $\boldsymbol{\theta}$ are a collection of $K$ multinomials $\langle \boldsymbol{\theta}_1, ..., \boldsymbol{\theta}_K \rangle$, the $k$th of which includes $N_k$ events. Note that there may be many derivations $\mathbf{y}$ for a given string $\mathbf{x}$—perhaps even infinitely many in some kinds of grammars. HMMs and vanilla PCFGs are the best known probabilistic grammars, but there are others. For example, in §5 we experiment with the "dependency model with valence," a probabilistic grammar for dependency parsing first proposed in [14].

## 3   Logistic Normal Prior on Probabilistic Grammars

A natural choice for a prior over the parameters of a probabilistic grammar is a Dirichlet prior. The Dirichlet family is conjugate to the multinomial family, which makes the inference more elegant and less computationally intensive. In addition, a Dirichlet prior can encourage sparse solutions, a property which is important with probabilistic grammars [11].

However, in [4], Blei and Lafferty noticed that the Dirichlet distribution is limited in its expressive power when modeling a corpus of documents, since it is less flexible about capturing relationships between possible topics. To solve this modeling issue, they extended the LDA model to use a logistic normal distribution [2] yielding *correlated* topic models. The logistic normal distribution maps a $d$-dimensional multivariate Gaussian to a distribution on the $d$-dimensional probability simplex, $S_d = \{\langle z_1, ..., z_d \rangle \in \mathbb{R}^d : z_i \geq 0, \sum_{i=1}^{d} z_i = 1\}$, by exponentiating the normally-distributed variables and normalizing.

Here we take a step analogous to Blei and Lafferty, aiming to capture correlations between the grammar's parameters. Our hierarchical generative model, which we call a *logistic-normal probabilistic grammar*, generates a sentence and derivation tree $\langle \mathbf{x}, \mathbf{y} \rangle$ as follows (see also Fig. 1):

1. Generate $\boldsymbol{\eta}_k \sim \mathcal{N}(\boldsymbol{\mu}_k, \Sigma_k)$ for $k = 1, ..., K$.

2. Set $\theta_{k,i} = \exp(\eta_{k,i}) \big/ \sum_{i'=1}^{N_k} \exp(\eta_{k,i'})$ for $k = 1, ..., K$ and $i = 1, ..., N_k$.

3. Generate $\mathbf{x}$ and $\mathbf{y}$ from $p(\mathbf{x}, \mathbf{y} \mid \boldsymbol{\theta})$ (i.e., sample from the probabilistic grammar).

We now turn to derive a variational inference algorithm for the model.[2] Variational Bayesian inference seeks an approximate posterior function $q(\boldsymbol{\eta}, \mathbf{y})$ which maximizes a lower bound (the negated variational free energy) on the log-likelihood [12], a bound which is achieved using Jensen's inequality:

$$\log p(\mathbf{x}, \mathbf{y} \mid \boldsymbol{\mu}, \boldsymbol{\Sigma}) \geq \sum_{i=1}^{K} \mathbb{E}_q \left[\log p(\boldsymbol{\eta}_i \mid \boldsymbol{\mu}_i, \Sigma_i)\right] + \mathbb{E}_q \left[\log p(\mathbf{x}, \mathbf{y} \mid \boldsymbol{\eta})\right] + H(q) \qquad (2)$$

We make a mean-field assumption, and assume that the posterior has the following form:

$$q(\boldsymbol{\eta}, \mathbf{y}) = \left( \prod_{k=1}^{K} \prod_{i=1}^{N_k} q(\eta_{k,i} \mid \tilde{\mu}_{k,i}, \tilde{\sigma}_{k,i}^2) \right) \times q(\mathbf{y}) \qquad (3)$$

where $q(\eta_{k,i} \mid \tilde{\mu}_{k,i}, \tilde{\sigma}_{k,i}^2)$ is a Gaussian $\mathcal{N}(\tilde{\mu}_{k,i}, \tilde{\sigma}_{k,i}^2)$.

Unfolding the expectation with respect to $q(\mathbf{y})$ in the second term in Eq. 2, while recalling that $\boldsymbol{\theta}$ is a deterministic function of $\boldsymbol{\eta}$, we have that:

$$\mathbb{E}_q \left[\log p(\mathbf{x}, \mathbf{y} \mid \boldsymbol{\eta})\right] = \mathbb{E}_{q(\boldsymbol{\eta})} \left[ \sum_{k=1}^{K} \sum_{i=1}^{N_k} \underbrace{\sum_{\mathbf{y}} q(\mathbf{y}) f_{k,i}(\mathbf{x}, \mathbf{y})}_{\tilde{f}_{k,i}} \log \theta_{k,i} \right]$$

$$= \mathbb{E}_{q(\boldsymbol{\eta})} \left[ \sum_{k=1}^{K} \sum_{i=1}^{N_k} \tilde{f}_{k,i} \left( \eta_{k,i} - \log \sum_{i'=1}^{N_k} \exp \eta_{k,i'} \right) \right] \qquad (4)$$

where $\tilde{f}_{k,i}$ is the expected number of occurrences of the $i$th event in distribution $k$, under $q(\mathbf{y})$.[3] The logarithm term in Eq. 4 is problematic, so we follow [4] in approximating it with a first-order Taylor expansion, introducing $K$ more variational parameters $\tilde{\zeta}_1, ..., \tilde{\zeta}_K$:

$$\log \left( \sum_{i'=1}^{N_k} \exp \eta_{k,i'} \right) \leq \log \tilde{\zeta}_k - 1 + \frac{1}{\tilde{\zeta}_k} \sum_{i'=1}^{N_k} \exp \eta_{k,i'} \qquad (5)$$

We now have

$$\mathbb{E}_q[\log p(\mathbf{x}, \mathbf{y} \mid \boldsymbol{\eta})] \geq \mathbb{E}_{q(\boldsymbol{\eta})} \left[ \sum_{k=1}^{K} \sum_{i=1}^{N_k} \tilde{f}_{k,i} \left( \eta_{k,i} - \log \tilde{\zeta}_k + 1 - \frac{1}{\tilde{\zeta}_k} \sum_{i'=1}^{N_k} \exp \eta_{k,i'} \right) \right] \quad (6)$$

$$= \sum_{k=1}^{K} \sum_{i=1}^{N_k} \tilde{f}_{k,i} \underbrace{\left( \tilde{\mu}_{k,i} - \log \tilde{\zeta}_k + 1 - \frac{1}{\tilde{\zeta}_k} \sum_{i'=1}^{N_k} \exp \left( \tilde{\mu}_{k,i} + \frac{\tilde{\sigma}_{k,i}^2}{2} \right) \right)}_{\tilde{\psi}_{k,i}}$$

$$= \sum_{k=1}^{K} \sum_{i=1}^{N_k} \tilde{f}_{k,i} \tilde{\psi}_{k,i} \qquad (7)$$

Note the shorthand $\tilde{\psi}_{k,i}$ to denote an expression involving $\tilde{\boldsymbol{\mu}}$, $\tilde{\boldsymbol{\sigma}}$, and $\tilde{\boldsymbol{\zeta}}$.

The final form of our bound is:[4]

$$\log p(\mathbf{x}, \mathbf{y} \mid \boldsymbol{\mu}, \boldsymbol{\Sigma}) \geq \left( \sum_{k=1}^{K} \mathbb{E}_q \left[ \log p(\boldsymbol{\eta}_k \mid \boldsymbol{\mu}_k, \Sigma_k) \right] \right) + \left( \sum_{k=1}^{K} \sum_{i=1}^{N_k} \tilde{f}_{k,i} \tilde{\psi}_{k,i} \right) + H(q) \quad (8)$$

Since, we are interested in EM-style algorithm, we will alternate between finding the maximizing $q(\boldsymbol{\eta})$ and the maximizing $q(\mathbf{y})$. Maximization with respect to $q(\boldsymbol{\eta})$ is not hard, because $q(\boldsymbol{\eta})$ is parametrized (see Appendix A). The following lemma shows that fortunately, finding the maximizing $q(\mathbf{y})$, which we did not parametrize originally, is not hard either:

**Lemma 1.** *Let $r(\mathbf{y} \mid \mathbf{x}, e^{\tilde{\psi}})$ denote the conditional distribution over $\mathbf{y}$ given $\mathbf{x}$ defined as:*

$$r(\mathbf{y} \mid \mathbf{x}, e^{\tilde{\psi}}) = \frac{1}{Z(\tilde{\boldsymbol{\psi}})} \prod_{k=1}^{K} \prod_{i=1}^{N_k} \exp \tilde{\psi}_{k,i} f_{k,i}(\mathbf{x}, \mathbf{y}) \quad (9)$$

*where $Z(\tilde{\boldsymbol{\psi}})$ is a normalization constant. Then $q(\mathbf{y}) = r(\mathbf{y} \mid \mathbf{x}, e^{\tilde{\psi}})$ maximizes the bound in Eq. 8.*

*Proof.* First note that $H(q) = H(q(\boldsymbol{\eta} \mid \tilde{\boldsymbol{\mu}}, \tilde{\boldsymbol{\sigma}})) + H(q(\mathbf{y}))$. This means that the terms we are interested in maximizing from Eq. 8 are the following, after writing down $\tilde{f}_{k,i}$ explicitly:

$$L = \operatorname*{argmax}_{q(\mathbf{y})} \sum_{\mathbf{y}} q(\mathbf{y}) \left( \sum_{k=1}^{K} \sum_{i=1}^{N_k} f_{k,i}(\mathbf{x}, \mathbf{y}) \tilde{\psi}_{k,i} \right) + H(q(\mathbf{y})) \quad (10)$$

However, note that:

$$L = \operatorname*{argmin}_{q(\mathbf{y})} D_{\mathrm{KL}}(q(\mathbf{y}) \| r(\mathbf{y} \mid \mathbf{x}, e^{\tilde{\psi}})) \quad (11)$$

where $D_{\mathrm{KL}}$ denotes the KL divergence. To see that, combine the definition of KL divergence with the fact that $\sum_{k=1}^{K} \sum_{i=1}^{N_k} f_{k,i}(\mathbf{x}, \mathbf{y}) \tilde{\psi}_{k,i} - \log Z(\tilde{\boldsymbol{\psi}}) = \log r(\mathbf{y} \mid \mathbf{x}, e^{\tilde{\psi}})$ where $\log Z(\tilde{\boldsymbol{\psi}})$ does not depend on $q(\mathbf{y})$. Eq. 11 is minimized when $q = r$. $\quad\square$

Interestingly, from the above lemma, the minimizing $q(\mathbf{y})$ has the same form as the probabilistic grammar in discussion, only without having sum-to-one constraints on $\boldsymbol{\theta}$ (leading to the required normalization constant). As in classic EM with probabilistic grammars, we never need to represent $q(\mathbf{y})$ explicitly; we need only $\tilde{\mathbf{f}}$, which can be calculated as expected feature values under $r(\mathbf{y} \mid \mathbf{x}, e^{\tilde{\psi}})$ using dynamic programming.

As noted, we are interested in a *point estimate* of $\boldsymbol{\theta}$. To achieve this, we will use the above variational method within an EM algorithm that estimates $\boldsymbol{\mu}$ and $\boldsymbol{\Sigma}$ in empirical Bayes fashion, then estimates $\boldsymbol{\theta}$ as $\boldsymbol{\mu}$, the mean of the learned prior. In the E-step, we maximize the bound with respect to the variational parameters ($\tilde{\boldsymbol{\mu}}$, $\tilde{\boldsymbol{\sigma}}$, $\tilde{\boldsymbol{\zeta}}$, and $\tilde{\mathbf{f}}$) using coordinate ascent. We optimize each of these separately in turn, cycling through, using appropriate optimization algorithms for each (conjugate gradient for $\tilde{\boldsymbol{\mu}}$, Newton's method for $\tilde{\boldsymbol{\sigma}}$, a closed form for $\tilde{\boldsymbol{\zeta}}$, and dynamic programming to solve for $\tilde{\mathbf{f}}$). In the M-step, we apply maximum likelihood estimation with respect to $\boldsymbol{\mu}$ and $\boldsymbol{\Sigma}$ given sufficient statistics gathered from the variational parameters in the E-step. The full algorithm is given in Appendix A.

## 4 Probabilistic Dependency Grammar Model

Dependency grammar [19] refers to linguistic theories that posit graphical representations of sentences in which words are vertices and the syntax is a tree. Such grammars can be context-free or context-sensitive in power, and they can be made probabilistic [9]. Dependency syntax is widely used in information extraction, machine translation, question

$$\mathbf{x} = \langle\text{NNP VBD JJ NNP}\rangle; \quad \mathbf{y} =$$ 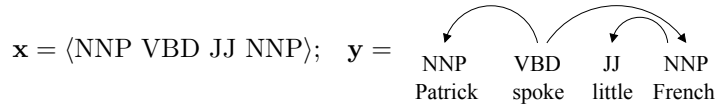

Figure 2: An example of a dependency tree (derivation $\mathbf{y}$). NNP denotes a proper noun, VBD a past-tense verb, and JJ an adjective, following the Penn Treebank conventions.

answering, and other natural language processing applications. Here, we are interested in *unsupervised* dependency parsing using the "dependency model with valence" [14]. The model is a probabilistic head automaton grammar [3] with a "split" form that renders inference cubic in the length of the sentence [6].

Let $\mathbf{x} = \langle x_1, x_2, ..., x_n\rangle$ be a sentence (here, as in prior work, represented as a sequence of part-of-speech tags). $x_0$ is a special "wall" symbol, \$, on the left of every sentence. A tree $\mathbf{y}$ is defined by a pair of functions $\mathbf{y}_{left}$ and $\mathbf{y}_{right}$ (both $\{0, 1, 2, ..., n\} \rightarrow 2^{\{1,2,...,n\}}$) that map each word to its sets of left and right dependents, respectively. Here, the graph is constrained to be a *projective* tree rooted at $x_0 = \$$: each word except \$ has a single parent, and there are no cycles or crossing dependencies. $\mathbf{y}_{left}(0)$ is taken to be empty, and $\mathbf{y}_{right}(0)$ contains the sentence's single head. Let $\mathbf{y}^{(i)}$ denote the subtree rooted at position $i$. The probability $P(\mathbf{y}^{(i)} \mid x_i, \boldsymbol{\theta})$ of generating this subtree, given its head word $x_i$, is defined recursively:

$$P(\mathbf{y}^{(i)} \mid x_i, \boldsymbol{\theta}) = \prod_{D \in \{left, right\}} \theta_{\text{s}}(\text{stop} \mid x_i, D, [\mathbf{y}_D(i) = \emptyset]) \tag{12}$$
$$\times \prod_{j \in \mathbf{y}_D(i)} \theta_{\text{s}}(\neg\text{stop} \mid x_i, D, \text{first}_{\mathbf{y}}(j)) \times \theta_{\text{c}}(x_j \mid x_i, D) \times P(\mathbf{y}^{(j)} \mid x_j, \boldsymbol{\theta})$$

where $\text{first}_{\mathbf{y}}(j)$ is a predicate defined to be true iff $x_j$ is the closest child (on either side) to its parent $x_i$. The probability of the entire tree is given by $p(\mathbf{x}, \mathbf{y} \mid \boldsymbol{\theta}) = P(\mathbf{y}^{(0)} \mid \$, \boldsymbol{\theta})$. The parameters $\boldsymbol{\theta}$ are the multinomial distributions $\theta_{\text{s}}(\cdot \mid \cdot, \cdot, \cdot)$ and $\theta_{\text{c}}(\cdot \mid \cdot, \cdot)$. To follow the general setting of Eq. 1, we index these distributions as $\boldsymbol{\theta}_1, ..., \boldsymbol{\theta}_K$. Figure 2 shows a dependency tree and its probability under this model.

## 5  Experiments

**Data**   Following the setting in [13], we experimented using part-of-speech sequences from the *Wall Street Journal* Penn Treebank [17], stripped of words and punctuation. We follow standard parsing conventions and train on sections 2–21,[5] tune on section 22, and report final results on section 23.

**Evaluation**   After learning a point estimate $\boldsymbol{\theta}$, we predict $\mathbf{y}$ for unseen test data (by parsing with the probabilistic grammar) and report the fraction of words whose predicted parent matches the gold standard corpus, known as attachment accuracy. Two parsing methods were considered: the most probable "Viterbi" parse ($\text{argmax}_{\mathbf{y}}\, p(\mathbf{y} \mid \mathbf{x}, \boldsymbol{\theta})$) and the minimum Bayes risk (MBR) parse ($\text{argmin}_{\mathbf{y}}\, \mathbb{E}_{p(\mathbf{y}'\mid\mathbf{x},\boldsymbol{\theta})}[\ell(\mathbf{y}; \mathbf{x}, \mathbf{y}')]$) with dependency attachment error as the loss function.

**Settings**   Our experiment compares four methods for estimating the probabilistic grammar's parameters:

**EM**   Maximum likelihood estimate of $\boldsymbol{\theta}$ using the EM algorithm to optimize $p(\mathbf{x} \mid \boldsymbol{\theta})$ [14].

**EM-MAP**   Maximum *a posteriori* estimate of $\boldsymbol{\theta}$ using the EM algorithm and a fixed symmetric Dirichlet prior with $\alpha > 1$ to optimize $p(\mathbf{x}, \boldsymbol{\theta} \mid \alpha)$. Tune $\alpha$ to maximize the likelihood of an unannotated development dataset, using grid search over $[1.1, 30]$.

**VB-Dirichlet** Use variational Bayes inference to estimate the posterior distribution $p(\boldsymbol{\theta} \mid \mathbf{x}, \alpha)$, which is a Dirichlet. Tune the symmetric Dirichlet prior's parameter $\alpha$ to maximize the likelihood of an unannotated development dataset, using grid search over $[0.0001, 30]$. Use the mean of the posterior Dirichlet as a point estimate for $\boldsymbol{\theta}$.

**VB-EM-Dirichlet** Use variational Bayes EM to optimize $p(\mathbf{x} \mid \boldsymbol{\alpha})$ with respect to $\boldsymbol{\alpha}$. Use the mean of the learned Dirichlet as a point estimate for $\boldsymbol{\theta}$ (similar to [5]).

**VB-EM-Log-Normal** Use variational Bayes EM to optimize $p(\mathbf{x} \mid \boldsymbol{\mu}, \boldsymbol{\Sigma})$ with respect to $\boldsymbol{\mu}$ and $\boldsymbol{\Sigma}$. Use the (exponentiated) mean of this Gaussian as a point estimate for $\boldsymbol{\theta}$.

Initialization is known to be important for EM as well as for the other algorithms we experiment with, since it involves non-convex optimization. We used the successful initializer from [14], which estimates $\boldsymbol{\theta}$ using soft counts on the training data where, in an $n$-length sentence, (a) each word is counted as the sentence's head $\frac{1}{n}$ times, and (b) each word $x_i$ attaches to $x_j$ proportional to $|i - j|$, normalized to a single attachment per word. This initializer is used with EM, EM-MAP, VB-Dirichlet, and VB-EM-Dirichlet. In the case of VB-EM-Log-Normal, it is used as an initializer both for $\boldsymbol{\mu}$ and inside the E-step. In all experiments reported here, we run the iterative estimation algorithm until the likelihood of a held-out, unannotated dataset stops increasing.

For learning with the logistic normal prior, we consider two initializations of the covariance matrices $\boldsymbol{\Sigma}_k$. The first is the $N_k \times N_k$ identity matrix. We then tried to bias the solution by injecting prior knowledge about the part-of-speech tags. Injecting a bias to parameter estimation of the DMV model has proved to be useful [18]. To do that, we mapped the tag set (34 tags) to twelve disjoint tag families.[6] The covariance matrices for all dependency distributions were initialized with 1 on the diagonal, 0.5 between tags which belong to the same family, and 0 otherwise. These results are given in Table 1 with the annotation "families."

**Results** Table 1 shows experimental results. We report attachment accuracy on three subsets of the corpus: sentences of length $\leq 10$ (typically reported in prior work and most similar to the training dataset), length $\leq 20$, and the full corpus. The Bayesian methods all outperform the common baseline (in which we attach each word to the word on its right), but the logistic normal prior performs considerably better than the other two methods as well.

The learned covariance matrices were very sparse when using the identity matrix to initialize. The diagonal values showed considerable variation, suggesting the importance of *variance* alone. When using the "tag families" initialization for the covariance, there were 151 elements across the covariance matrices which were not identically 0 (out of more than 1,000), pointing to a learned relationship between parameters. In this case, most covariance matrices for $\theta_{\mathrm{c}}$ dependencies were diagonal, while many of the covariance matrices for the stopping probabilities ($\theta_{\mathrm{s}}$) had significant correlations.

## 6 Conclusion

We have considered a Bayesian model for probabilistic grammars, which is based on the logistic normal prior. Experimentally, several different approaches for grammar induction were compared based on different priors. We found that a logistic normal prior outperforms earlier approaches, presumably because it can capitalize on similarity between part-of-speech tags, as different tags tend to appear as arguments in similar syntactic contexts. We achieved state-of-the-art unsupervised dependency parsing results.

| | attachment accuracy (%) | | | | | |
| --- | --- | --- | --- | --- | --- | --- |
| | Viterbi decoding | | | MBR decoding | | |
| | $\lvert\mathbf{x}\rvert \leq 10$ | $\lvert\mathbf{x}\rvert \leq 20$ | all | $\lvert\mathbf{x}\rvert \leq 10$ | $\lvert\mathbf{x}\rvert \leq 20$ | all |
| Attach-Right | 38.4 | 33.4 | 31.7 | 38.4 | 33.4 | 31.7 |
| EM | 45.8 | 39.1 | 34.2 | 46.1 | 39.9 | 35.9 |
| EM-MAP, $\boldsymbol{\alpha} = 1.1$ | 45.9 | 39.5 | 34.9 | 46.2 | 40.6 | 36.7 |
| VB-Dirichlet, $\boldsymbol{\alpha} = 0.25$ | 46.9 | 40.0 | 35.7 | 47.1 | 41.1 | 37.6 |
| VB-EM-Dirichlet | 45.9 | 39.4 | 34.9 | 46.1 | 40.6 | 36.9 |
| VB-EM-Log-Normal, $\boldsymbol{\Sigma}_k^{(0)} = \mathbf{I}$ | 56.6 | 43.3 | 37.4 | 59.1 | **45.9** | 39.9 |
| VB-EM-Log-Normal, families | **59.3** | **45.1** | **39.0** | 59.4 | 45.9 | 40.5 |

Table 1: Attachment accuracy of different learning methods on unseen test data from the Penn Treebank of varying levels of difficulty imposed through a length filter. Attach-Right attaches each word to the word on its right and the last word to $. EM and EM-MAP with a Dirichlet prior ($\alpha > 1$) are reproductions of earlier results [14, 18].

## Acknowledgments

The authors would like to thank the anonymous reviewers, John Lafferty, and Matthew Harrison for their useful feedback and comments. This work was made possible by an IBM faculty award, NSF grants IIS-0713265 and IIS-0836431 to the third author and computational resources provided by Yahoo.

## Footnotes

[1]A certain variant of LDA can be seen as a Bayesian version of a zero-order HMM, where the unigram state (topic) distribution is sampled first for each sequence (document).

[2]We note that variational inference algorithms have been successfully applied to grammar learning tasks, for example, in [16] and [15].

[3]With probabilistic grammars, this quantity can be computed using a summing dynamic programming algorithm like the forward-backward or inside-outside algorithm.

[4]A tighter bound was proposed in [1], but we follow [4] for simplicity.

[5]Training in the unsupervised setting for this data set can be expensive, and requires running a cubic-time dynamic programming algorithm iteratively, so we follow common practice in restricting the training set (but not development or test sets) to sentences of length ten or fewer words. Short sentences are also less structurally ambiguous and may therefore be easier to learn from.

[6]These are simply coarser tags: adjective, adverb, conjunction, foreign, interjection, noun, number, particle, preposition, pronoun, proper, verb. The coarse tags were chosen manually to fit seven treebanks in different languages.

[7]An implementation of the algorithm is available at `http://www.ark.cs.cmu.edu/DAGEEM`.

## References

[1] A. Ahmed and E. Xing. On tight approximate inference of the logistic normal topic admixture model. In *Proc. of AISTATS*, 2007.

[2] J. Aitchison and S. M. Shen. Logistic-normal distributions: some properties and uses. *Biometrika*, 67:261–272, 1980.

[3] H. Alshawi and A. L. Buchsbaum. Head automata and bilingual tiling: Translation with minimal representations. In *Proc. of ACL*, 1996.

[4] D. Blei and J. D. Lafferty. Correlated topic models. In *Proc. of NIPS*, 2006.

[5] D. Blei, A. Ng, and M. Jordan. Latent Dirichlet allocation. *Journal of Machine Learning Research*, 3:993–1022, 2003.

[6] J. Eisner. Bilexical grammars and a cubic-time probabilistic parser. In *Proc. of IWPT*, 1997.

[7] J. Eisner. Transformational priors over grammars. In *Proc. of EMNLP*, 2002.

[8] J. R. Finkel, C. D. Manning, and A. Y. Ng. Solving the problem of cascading errors: Approximate Bayesian inference for linguistic annotation pipelines. In *Proc. of EMNLP*, 2006.

[9] H. Gaifman. Dependency systems and phrase-structure systems. *Information and Control*, 8, 1965.

[10] S. Goldwater and T. L. Griffiths. A fully Bayesian approach to unsupervised part-of-speech tagging. In *Proc. of ACL*, 2007.

[11] M. Johnson, T. L. Griffiths, and S. Goldwater. Bayesian inference for PCFGs via Markov chain Monte Carlo. In *Proc. of NAACL*, 2007.

[12] M. I. Jordan, Z. Ghahramani, T. S. Jaakola, and L. K. Saul. An introduction to variational methods for graphical models. *Machine Learning*, 37(2):183–233, 1999.

[13] D. Klein and C. D. Manning. A generative constituent-context model for improved grammar induction. In *Proc. of ACL*, 2002.

[14] D. Klein and C. D. Manning. Corpus-based induction of syntactic structure: Models of dependency and constituency. In *Proc. of ACL*, 2004.

[15] K. Kurihara and T. Sato. Variational Bayesian grammar induction for natural language. In *Proc. of ICGI*, 2006.

[16] P. Liang, S. Petrov, M. Jordan, and D. Klein. The infinite PCFG using hierarchical Dirichlet processes. In *Proc. of EMNLP*, 2007.

[17] M. P. Marcus, B. Santorini, and M. A. Marcinkiewicz. Building a large annotated corpus of English: The Penn treebank. *Computational Linguistics*, 19:313–330, 1993.

[18] N. A. Smith and J. Eisner. Annealing structural bias in multilingual weighted grammar induction. In *Proc. of COLING-ACL*, 2006.

[19] L. Tesnière. *Élément de Syntaxe Structurale*. Klincksieck, 1959.

[20] K. Toutanova and M. Johnson. A Bayesian LDA-based model for semi-supervised part-of-speech tagging. In *Proc. of NIPS*, 2007.

# A   VB-EM for Logistic-Normal Probabilistic Grammars

The algorithm for variational inference with probabilistic grammars using logistic normal prior follows.[7] Since the updates for $\tilde{\zeta}_k^{l,(t)}$ are fast, we perform them after each optimization routine in the E-step (suppressed for clarity). There are variational parameters for each training example, indexed by $\ell$. We denote by $B$ the variational bound in Eq. 8. Our stopping criterion relies on the likelihood of a held-out set (§5) using a point estimate of the model.

**Input**: initial parameters $\boldsymbol{\mu}^{(0)}$, $\boldsymbol{\Sigma}^{(0)}$, training data $\mathbf{x}$, and development data $\mathbf{x}'$
**Output**: learned parameters $\boldsymbol{\mu}$, $\boldsymbol{\Sigma}$
$t \leftarrow 1$ ;
**repeat**

    *E-step (for each training example $\ell = 1, ..., M$):*   **repeat**

        optimize for $\tilde{\mu}_k^{\ell,(t)}, k = 1, ..., K$: use conjugate gradient descent with

        $\frac{\partial L}{\partial \tilde{\mu}_{k,i}^{\ell}} = -\left( (\Sigma_k^{(t-1)})^{-1})(\mu_k^{(t-1)} - \tilde{\mu}_k^{\ell}) \right)_i - \tilde{f}_{k,i}^{\ell} + \sum_{i'=1}^{N_k} \left( \tilde{f}_{k,i'}/\tilde{\zeta}_k \right) \exp \left( \tilde{\mu}_{k,i'} + \tilde{\sigma}_{k,i'}^2/2 \right)$;

        optimize $\tilde{\sigma}_k^{\ell,(t)}, k = 1, ..., K$: use Newton's method for each coordinate (with $\tilde{\sigma}_{k,i}^{\ell} > 0$) with

        $\frac{\partial L}{\partial \tilde{\sigma}_{k,i}^2} = -\Sigma_{k,ii}^{(t-1)}/2 - \left( \sum_{i'=1}^{N_k} \tilde{f}_{k,i'} \right) \exp(\tilde{\mu}_{k,i} + \tilde{\sigma}_{k,i}^2/2)/2\tilde{\zeta}_k + 1/2\tilde{\sigma}_{k,i}^2$;

        update $\tilde{\zeta}_k^{\ell,(t)}, \forall k$: $\tilde{\zeta}_k^{\ell,(t)} \leftarrow \sum_{i=1}^{N_k} \exp \left( \tilde{\mu}_{k,i}^{\ell,(t)} + (\tilde{\sigma}_{k,i}^{\ell,(t)})^2/2 \right)$ ;

        update $\tilde{\boldsymbol{\psi}}_k^{\ell,(t)}, \forall k$: $\tilde{\psi}_{k,i}^{\ell,(t)} \leftarrow \tilde{\mu}_{k,i}^{\ell,(t)} - \log \tilde{\zeta}_k^{\ell,(t)} + 1 - \frac{1}{\tilde{\zeta}_k^{\ell,(t)}} \sum_{i'=1}^{N_k} \exp \left( \tilde{\mu}_{k,i}^{\ell,(t)} + (\tilde{\sigma}_{k,i}^{\ell,(t)})^2/2 \right)$ ;

        compute expected counts $\tilde{\mathbf{f}}_k^{\ell,(t)}, k = 1, ..., K$: use an inside-outside algorithm to re-estimate expected counts $\tilde{f}_{k,i}^{\ell,(t)}$ in weighted grammar $q(y)$ with weights $e^{\tilde{\psi}^{\ell}}$ ;

    **until** *B does not change* ;

    *M-step*: Estimate $\boldsymbol{\mu}^{(t)}$ and $\boldsymbol{\Sigma}^{(t)}$ using the following maximum likelihood closed form solution:

$$\mu_{k,i}^{(t)} \quad \leftarrow \quad \frac{1}{M} \sum_{\ell=1}^M \tilde{\mu}_{k,i}^{\ell,(t)}$$

$$\left[ \Sigma_k^{(t)} \right]_{i,j} \quad \leftarrow \quad \frac{1}{M} \left( \sum_{\ell=1}^M \tilde{\mu}_{k,i}^{\ell,(t)} \tilde{\mu}_{k,j}^{\ell,(t)} + (\tilde{\sigma}^{\ell,(t)})_{k,i}^2 \delta_{i,j} + M\mu_{k,i}^{(t)}\mu_{k,j}^{(t)} - \mu_{k,j}^{(t)} \sum_{\ell=1}^M \tilde{\mu}_{k,i}^{\ell,(t)} - \mu_{k,i}^{(t)} \sum_{\ell=1}^M \tilde{\mu}_{k,j}^{\ell,(t)} \right)$$

    where $\delta_{i,j} = 1$ if $i = j$ and 0 otherwise.
**until** *likelihood of held-out data, $p(\mathbf{x}' \mid E[\boldsymbol{\mu}^{(t)}])$, decreases* ;
$t \leftarrow t + 1$;
**return** $\boldsymbol{\mu}^{(t)}$, $\boldsymbol{\Sigma}^{(t)}$

